# The Intelligent Surfer: Probabilistic Combination of Link and Content Information in PageRank

**Matthew Richardson**      **Pedro Domingos**
Department of Computer Science and Engineering
University of Washington
Box 352350
Seattle, WA 98195-2350, USA
*{mattr, pedrod}@cs.washington.edu*

## Abstract

The PageRank algorithm, used in the Google search engine, greatly improves the results of Web search by taking into account the link structure of the Web. PageRank assigns to a page a score proportional to the number of times a random surfer would visit that page, if it surfed indefinitely from page to page, following all outlinks from a page with equal probability. We propose to improve PageRank by using a more intelligent surfer, one that is guided by a probabilistic model of the relevance of a page to a query. Efficient execution of our algorithm at query time is made possible by precomputing at crawl time (and thus once for all queries) the necessary terms. Experiments on two large subsets of the Web indicate that our algorithm significantly outperforms PageRank in the (human-rated) quality of the pages returned, while remaining efficient enough to be used in today's large search engines.

## 1   Introduction

Traditional information retrieval techniques can give poor results on the Web, with its vast scale and highly variable content quality. Recently, however, it was found that Web search results can be much improved by using the information contained in the link structure between pages. The two best-known algorithms which do this are HITS [1] and PageRank [2]. The latter is used in the highly successful Google search engine [3]. The heuristic underlying both of these approaches is that pages with many inlinks are more likely to be of high quality than pages with few inlinks, given that the author of a page will presumably include in it links to pages that s/he believes are of high quality. Given a query (set of words or other query terms), HITS invokes a traditional search engine to obtain a set of pages relevant to it, expands this set with its inlinks and outlinks, and then attempts to find two types of pages, *hubs* (pages that point to many pages of high quality) and *authorities* (pages of high quality). Because this computation is carried out at query time, it is not feasible for today's search engines, which need to handle tens of millions of queries per day. In contrast, PageRank computes a single measure of quality for a page at crawl time. This meas-

ure is then combined with a traditional information retrieval score at query time. Compared with HITS, this has the advantage of much greater efficiency, but the disadvantage that the PageRank score of a page ignores whether or not the page is relevant to the query at hand.

Traditional information retrieval measures like TFIDF [4] rate a document highly if the query terms occur frequently in it. PageRank rates a page highly if it is at the center of a large sub-web (i.e., if many pages point to it, many other pages point to those, etc.). Intuitively, however, the best pages should be those that are at the center of a large sub-web *relevant to the query*. If one issues a query containing the word *jaguar*, then pages containing the word *jaguar* that are also pointed to by many other pages containing *jaguar* are more likely to be good choices than pages that contain *jaguar* but have no inlinks from pages containing it. This paper proposes a search algorithm that formalizes this intuition while, like PageRank, doing most of its computations at crawl time. The PageRank score of a page can be viewed as the rate at which a surfer would visit that page, if it surfed the Web indefinitely, blindly jumping from page to page. Our algorithm does something closer to what a human surfer would do, jumping preferentially to pages containing the query terms.

A problem common to both PageRank and HITS is topic drift. Because they give the same weight to all edges, the pages with the most inlinks in the network being considered (either at crawl or query time) tend to dominate, whether or not they are the most relevant to the query. Chakrabarti et al. [5] and Bharat and Henzinger [6] propose heuristic methods for differentially weighting links. Our algorithm can be viewed as a more principled approach to the same problem. It can also be viewed as an analog for PageRank of Cohn and Hofmann's [7] variation of HITS. Rafiei and Mendelzon's [8] algorithm, which biases PageRank towards pages containing a specific word, is a predecessor of our work. Haveliwala [9] proposes applying an optimized version of PageRank to the subset of pages containing the query terms, and suggests that users do this on their own machines.

We first describe PageRank. We then introduce our query-dependent, content-sensitive version of PageRank, and demonstrate how it can be implemented efficiently. Finally, we present and discuss experimental results.

## 2   PageRank : The Random Surfer

Imagine a web surfer who jumps from web page to web page, choosing with uniform probability which link to follow at each step. In order to reduce the effect of dead-ends or endless cycles the surfer will occasionally jump to a random page with some small probability $\beta$, or when on a page with no out-links. To reformulate this in graph terms, consider the web as a directed graph, where nodes represent web pages, and edges between nodes represent links between web pages. Let $\mathbf{W}$ be the set of nodes, $N=|\mathbf{W}|$, $\mathbf{F}_i$ be the set of pages page $i$ links to, and $\mathbf{B}_i$ be the set pages which link to page $i$. For pages which have no outlinks we add a link to all pages in the graph[1]. In this way, rank which is lost due to pages with no outlinks is redistributed uniformly to all pages. If averaged over a sufficient number of steps, the probability the surfer is on page $j$ at some point in time is given by the formula:

$$P(j) = \frac{(1-\beta)}{N} + \beta \sum_{i \in B_j} \frac{P(i)}{|F_i|} \tag{1}$$

The PageRank score for node $j$ is defined as this probability: $PR(j)=P(j)$. Because equation (1) is recursive, it must be iteratively evaluated until $P(j)$ converges. Typically, the initial distribution for $P(j)$ is uniform. PageRank is equivalent to the primary eigenvector of the transition matrix Z:

$$Z = (1-\beta)\left[\frac{1}{N}\right]_{NxN} + \beta M \text{ ,with } \quad M_{ji} = \begin{cases} \dfrac{1}{|F_i|} & \text{if there is an edge from } i \text{ to } j \\ 0 & \text{otherwise} \end{cases} \quad (2)$$

One iteration of equation (1) is equivalent to computing $\mathbf{x}^{t+1}=Z\mathbf{x}^t$, where $\mathbf{x}_j^t=P(j)$ at iteration $t$. After convergence, we have $\mathbf{x}^{T+1}=\mathbf{x}^T$, or $\mathbf{x}^T=Z\mathbf{x}^T$, which means $\mathbf{x}^T$ is an eigenvector of Z. Furthermore, since the columns of Z are normalized, $\mathbf{x}$ has an eigenvalue of 1.

## 3 Directed Surfer Model

We propose a more intelligent surfer, who probabilistically hops from page to page, depending on the content of the pages and the query terms the surfer is looking for. The resulting probability distribution over pages is:

$$P_q(j) = (1-\beta)P_q'(j) + \beta \sum_{i \in B_j} P_q(i)P_q(i \to j) \quad (3)$$

where $P_q(i{\to}j)$ is the probability that the surfer transitions to page $j$ given that he is on page $i$ and is searching for the query $q$. $P_q'(j)$ specifies where the surfer chooses to jump when not following links. $P_q(j)$ is the resulting probability distribution over pages and corresponds to the *query-dependent PageRank* score (QD-PageRank$_q$(j) $\equiv$ $P_q(j)$). As with PageRank, QD-PageRank is determined by iterative evaluation of equation 3 from some initial distribution, and is equivalent to the primary eigenvector of the transition matrix Z$_q$, where $Z_{q_{ji}} = (1-\beta)P_q'(j) + \beta \sum_{i \in B_j} P_q(i \to j)$ . Although

$P_q(i{\to}j)$ and $P_q'(j)$ are arbitrary distributions, we will focus on the case where both probability distributions are derived from $R_q(j)$, a measure of *relevance* of page $j$ to query $q$:

$$P_q'(j) = \frac{R_q(j)}{\sum_{k \in W} R_q(k)} \qquad\qquad P_q(i \to j) = \frac{R_q(j)}{\sum_{k \in F_i} R_q(k)} \qquad (4)$$

In other words, when choosing among multiple out-links from a page, the directed surfer tends to follow those which lead to pages whose content has been deemed relevant to the query (according to $R_q$). Similarly to PageRank, when a page's out-links all have zero relevance, or has no outlinks, we add links from that page to all other pages in the network. On such a page, the surfer thus chooses a new page to jump to according to the distribution $P_q'(j)$.

When given a multiple-term query, $Q=\{q_1,q_2,...\}$, the surfer selects a $q$ according to some probability distribution, $P(q)$ and uses that term to guide its behavior (according to equation 3) for a large number of steps[1]. It then selects another term according to the distribution to determine its behavior, and so on. The resulting distribution over visited web pages is QD-PageRank$_Q$ and is given by

$$\text{QD} - \text{PageRank}_Q(j) \equiv P_Q(j) = \sum_{q \in Q} P(q) P_q(j) \qquad (5)$$

For standard PageRank, the PageRank vector is equivalent to the primary eigenvector of the matrix Z. The vector of single-term QD-PageRank$_q$ is again equivalent to the primary eigenvector of the matrix $Z_q$. An interesting question that arises is whether the QD-PageRank$_Q$ vector is equivalent to the primary eigenvector of a matrix $\mathbf{Z}_Q = \sum_{q \in Q} P(q) \mathbf{Z}_q$ (corresponding to the combination performed by equation 5). In fact, this is not the case. Instead, the primary eigenvector of $\mathbf{Z}_Q$ corresponds to the QD-PageRank obtained by a random surfer who, *at each step*, selects a new query according to the distribution $P(q)$. However, QD-PageRank$_Q$ is approximately equal to the PageRank that results from this single-step surfer, for the following reason.

Let $\mathbf{x}_q$ be the L2-normalized primary eigenvector for matrix $\mathbf{Z}_q$ (note element $j$ of $\mathbf{x}_q$ is QD-PageRank$_q(j)$), thus satisfying $\mathbf{x}^i = \mathbf{T}^i \mathbf{x}^i$. Since $\mathbf{x}_q$ is the primary eigenvector for $\mathbf{Z}_q$, we have [10]: $\forall q, r \in Q : \left\| \mathbf{Z}_q \mathbf{x}_q \right\| \geq \left\| \mathbf{Z}_q \mathbf{x}_r \right\|$. Thus, to a first degree of approximation, $\mathbf{Z}_q \sum_{r \in Q} \mathbf{x}_r \approx \kappa \mathbf{Z}_q \mathbf{x}_q$. Suppose $P(q) = 1/|Q|$. Consider $\mathbf{x}_Q = \sum_{q \in Q} P(q) \mathbf{x}_q$ (see equation 5). Then

$$\mathbf{Z}_Q \mathbf{x}_Q = \left( \sum_{q \in Q} \frac{1}{|Q|} \mathbf{Z}_q \right) \left( \sum_{q \in Q} \mathbf{x}_q \right) = \frac{1}{|Q|} \sum_{q \in Q} \left( \mathbf{Z}_q \sum_{r \in Q} \mathbf{x}_r \right) \approx \frac{1}{|Q|} \sum_{q \in Q} \left( \kappa \mathbf{Z}_q \mathbf{x}_q \right) = \frac{\kappa}{|Q|} \sum_{q \in Q} \mathbf{x}_q = \frac{\kappa}{n} \mathbf{x}_Q$$

and thus $\mathbf{x}_Q$ is approximately an eigenvector for $\mathbf{Z}_Q$. Since $\mathbf{x}_Q$ is equivalent to QD-PageRank$_Q$, and $\mathbf{Z}_Q$ describes the behavior of the single-step surfer, QD-PageRank$_Q$ is approximately the same PageRank that would be obtained by using the single-step surfer. The approximation has the least error when the individual random surfers defined by $\mathbf{Z}_q$ are very similar, or are very dissimilar.

The choice of relevance function $R_q(j)$ is arbitrary. In the simplest case, $R_q(j)=R$ is independent of the query term and the document, and QD-PageRank reduces to PageRank. One simple content-dependent function could be $R_q(j)=1$ if the term q appears on page *j*, and 0 otherwise. Much more complex functions could be used, such as the well-known TFIDF information retrieval metric, a score obtained by latent semantic indexing, or any heuristic measure using text size, positioning, etc…. It is important to note that most current text ranking functions could be easily incorporated into the directed surfer model.

## 4 Scalability

The difficulty with calculating a query dependent PageRank is that a search engine cannot perform the computation, which can take hours, at query time, when it is expected to return results in seconds (or less). We surmount this problem by precomputing the individual term rankings QD-PageRank$_q$, and combining them at query time according to equation 5. We show that the computation and storage requirements for QD-PageRank$_q$ for hundreds of thousands of words is only approximately 100-200 times that of a single query independent PageRank.

Let $W=\{q_1, q_2, …, q_m\}$ be the set of words in our lexicon. That is, we assume all search queries contain terms in W, or we are willing to use plain PageRank for those terms not in W. Let $d_q$ be the number of documents which contain the term q. Then $S = \sum_{q \in W} d_q$ is the number of unique document-term pairs.

## 4.1 Disk Storage

For each term $q$, we must store the results of the computation. We add the minor restriction that a search query will only return documents containing all of the terms[1]. Thus, when merging QD-PageRank$_q$'s, we need only to know the QD-PageRank$_q$ for documents that contain the term. Each QD-PageRank$_q$ is a vector of $d_q$ values. Thus, the space required to store all of the PageRanks is S, a factor of S/N times the query independent PageRank alone (recall N is the number of web pages). Further, note that the storage space is still considerably less than that required for the search engine's reverse index, which must store information about all document-term pairs, as opposed to our need to store information about every *unique* document term pair.

## 4.2 Time Requirements

If $R_q(j)=0$ for some document j, the directed surfer will never arrive at that page. In this case, we know QD-PageRank$_q$(j)=0, and thus when calculating QD-PageRank$_q$, we need only consider the subset of nodes for which $R_q(j)>0$. We add the reasonable constraint that $R_q(j)=0$ if term $q$ does not appear in document $j$, which is common for most information retrieval relevance metrics, such as TFIDF. The computation for term $q$ then only needs to consider $d_q$ documents. Because it is proportional to the number of documents in the graph, the computation of QD-PageRank$_q$ for all q in W will require O(S) time, a factor of S/N times the computation of the query independent PageRank alone. Furthermore, we have noticed in our experiments that the computation converges in fewer iterations on these smaller sub-graphs, empirically reducing the computational requirements to 0.75*S/N. Additional speedup may be derived from the fact that for most words, the sub-graph will completely fit in memory, unlike PageRank which (for any large corpus) must repeatedly read the graph structure from disk during computation.

## 4.3 Empirical Scalability

The fraction S/N is critical to determining the scalability of QD-PageRank. If every document contained vastly different words, S/N would be proportional to the number of search terms, m. However, this is not the case. Instead, there are a very few words that are found in almost every document, and many words which are found in very few documents[2]; in both cases the contribution to S is small.

In our database of 1.7 million pages (see section 5), we let W be the set of all unique words, and removed the 100 most common words[3]. This results in |W|=2.3 million words, and the ratio S/N was found to be 165. We expect that this ratio will remain relatively constant even for much larger sets of web pages. This means QD-PageRank requires approximately 165 times the storage space and 124 times the computation time to allow for arbitrary queries over any of the 2.3 million words (which is still less storage space than is required by the search engine's reverse index alone).

| Table 1: Results on *educrawl* | | | Table 2: Results on *WebBase* | | |
|---|---|---|---|---|---|
| Query | QD-PR | PR | Query | QD-PR | PR |
| chinese association | 10.75 | 6.50 | alcoholism | 11.50 | 11.88 |
| computer labs | 9.50 | 13.25 | architecture | 8.45 | 2.93 |
| financial aid | 8.00 | 12.38 | bicycling | 8.45 | 6.88 |
| intramural | 16.5 | 10.25 | rock climbing | 8.43 | 5.75 |
| maternity | 12.5 | 6.75 | shakespeare | 11.53 | 5.03 |
| president office | 5.00 | 11.38 | stamp collecting | 9.13 | 10.68 |
| sororities | 13.75 | 7.38 | vintage car | 13.15 | 8.68 |
| student housing | 14.13 | 10.75 | Thailand tourism | 16.90 | 9.75 |
| visitor visa | 19.25 | 12.50 | Zen Buddhism | 8.63 | 10.38 |
| **Average** | **12.15** | **10.13** | **Average** | **10.68** | **7.99** |

## 5  Results

We give results on two data sets: *educrawl*, and *WebBase*. *Educrawl* is a crawl of the web, restricted to .edu domains. The crawler was seeded with the first 18 results of a search for "University" on Google (www.google.com). Links containing "?" or "cgi-bin" were ignored, and links were only followed if they ended with ".html". The crawl contains 1.76 million pages over 32,000 different domains. *WebBase* is the first 15 million pages of the Stanford WebBase repository [12], which contains over 120 million pages. For both datasets, HTML tags were removed before processing.

We calculated QD-PageRank as described above, using $R_q(j)$ = the fraction of words equal to $q$ in page $j$, and $P(q)=1/|Q|$. We compare our algorithm to the standard PageRank algorithm. For content ranking, we used the same $R_q(j)$ function as for QD-PageRank, but, similarly to TFIDF, weighted the contribution of each search term by the log of its inverse document frequency. As there is nothing published about merging PageRank and content rank into one list, the approach we follow is to normalize the two scores and add them. This implicitly assumes that PageRank and content rank are equally important. This resulted in poor PageRank performance, which we found was because the distribution of PageRanks is much more skewed than the distribution of content ranks; normalizing the vectors resulted in PageRank primarily determining the final ranking. To correct this problem, we scaled each vector to have the same average value in its top ten terms before adding the two vectors. This drastically improved PageRank.

For *educrawl*, we requested a single word and two double word search queries from each of three volunteers, resulting in a total of nine queries. For each query, we randomly mixed the top 10 results from standard PageRank with the top 10 results from QD-PageRank, and gave them to four volunteers, who were asked to rate each search result as a 0 (not relevant), 1 (somewhat relevant, not very good), or 2 (good search result) based on the contents of the page it pointed to. In Table 1, we present the final rating for each method, per query. This rating was obtained by first summing the ratings for the ten pages from each method for each volunteer, and then averaging the individual ratings. A similar experiment for *WebBase* is given in Table 2. For *WebBase*, we randomly selected the queries from Bharat and Henzinger [6]. The four volunteers for the *WebBase* evaluation were independent from the four for the *educrawl* evaluation, and none knew how the pages they were asked to rate were obtained.

QD-PageRank performs better than PageRank, accomplishing a relative improvement in relevance of 20% on *educrawl* and 34% on *WebBase*. The results are statistically significant (p<.03 for educrawl and p<.001 for WebBase using a two-tailed paired t-test, one sample per person per query). Averaging over queries, every volunteer found QD-PageRank to be an improvement over PageRank, though not all differences were statistically significant.

One item to note is that the results on multiple word queries are not as positive as the results on single word queries. As discussed in section 3, the combination of single word QD-PageRanks to calculate the QD-PageRank for a multiple word query is only an approximation, made for practical reasons. This approximation is worse when the words are highly dependent. Further, some queries, such as "financial aid" have a different intended meaning as a phrase than simply the two words "financial" and "aid". For queries such as these, the words are highly dependent. We could partially overcome this difficulty by adding the most common phrases to the lexicon, thus treating them the same as single words.

# 6   Conclusions

In this paper, we introduced a model that probabilistically combines page content and link structure in the form of an intelligent random surfer. The model can accommodate essentially any query relevance function in use today, and produces higher-quality results than PageRank, while having time and storage requirements that are within reason for today's large scale search engines.

### Acknowledgments

We would like to thank Gary Wesley and Taher Haveliwala for their help with Web-Base, Frank McSherry for eigen-help, and our experiment volunteers for their time. This work was partially supported by NSF CAREER and IBM Faculty awards to the second author.

## Footnotes

[1] For each page $s$ with no outlinks, we set $\mathbf{F}_s=\{$all N nodes$\}$, and for all other nodes augment $\mathbf{B}_i$ with $s$. ($\mathbf{B}_i \cup \{s\}$)

[1] However many steps are needed to reach convergence of equation 3.

[1] Google has this "feature" as well. See http://www.google.com/technology/whyuse.html.

[2] This is because the distribution of words in text tends to follow an inverse power law [11]. We also verified experimentally that the same holds true for the distribution of the number of documents a word is found in.

[3] It is common to remove "stop" words such as *the*, *is,* etc., as they do not affect the search.

### References

[1] J. M. Kleinberg (1998). Authoritative sources in a hyperlinked environment. *Proceedings of the Ninth Annual ACM-SIAM Symposium on Discrete Algorithms*.

[2] L. Page, S. Brin, R. Motwani, and T. Winograd (1998). The PageRank citation ranking: Bringing order to the web. Technical report, Stanford University, Stanford, CA.

[3] S. Brin and L. Page (1998). The anatomy of a large-scale hypertextual Web search engine. *Proceedings of the Seventh International World Wide Web Conference*.

[4] G. Salton and M. J. McGill (1983). *Introduction to Modern Information Retrieval*. McGraw-Hill, New York, NY.

[5] S. Chakrabarti, B. Dom, D. Gibson, J. Kleinberg, P. Raghavan, and S. Rajagopalan (1998). Automatic resource compilation by analyzing hyperlink structure and associated text. *Proceedings of the Seventh International World Wide Web Conference*.

[6] K. Bharat and M. R. Henzinger (1998). Improved algorithms for topic distillation in a hyperlinked environment. *Proceedings of the Twenty-First Annual International ACM SIGIR Conference on Research and Development in Information Retrieval*.

[7] D. Cohn and T. Hofmann (2001). The missing link - a probabilistic model of document content and hypertext connectivity. In T. K. Leen, T. G. Dietterich, and V. Tresp, editors, *Advances in Neural Information Processing Systems 13*. MIT Press, Cambridge, MA.

[8] D. Rafiei and A. Mendelzon (2000). What is this page known for? Computing web page reputations. *Proceedings of the Ninth International World Wide Web Conference*.

[9] T. Haveliwala (1999). Efficient computation of PageRank. Technical report, Stanford University, Stanford, CA.

[10] G. H. Golub and C. F. Van Loan (1996). *Matrix Computations*. Johns Hopkins University Press, Baltimore, MD, third edition.

[11] G. K. Zipf (1949). *Human Behavior and the Principle of Least Effort*. Addison-Wesley, Cambridge, MA.

[12] J. Hirai, S. Raghaven, H. Garcia-Molina, A. Paepcke (1999). WebBase: a repository of web pages. *Proceedings of the Ninth World Wide Web Conference*.
